# A model of transparent motion and non-transparent motion aftereffects

**Alexander Grunewald***

Max-Planck Institut für biologische Kybernetik
Spemannstraße 38
D-72076 Tübingen, Germany

## Abstract

A model of human motion perception is presented. The model contains two stages of direction selective units. The first stage contains broadly tuned units, while the second stage contains units that are narrowly tuned. The model accounts for the motion aftereffect through adapting units at the first stage and inhibitory interactions at the second stage. The model explains how two populations of dots moving in *slightly* different directions are perceived as a single population moving in the direction of the vector sum, and how two populations moving in *strongly* different directions are perceived as transparent motion. The model also explains why the motion aftereffect in both cases appears as non-transparent motion.

## 1 INTRODUCTION

Transparent motion can be studied using displays which contain two populations of moving dots. The dots within each population have the same direction of motion, but directions can differ between the two populations. When the two directions are very similar, subjects report seeing dots moving in the average direction (Williams & Sekuler, 1984). However, when the difference between the two directions gets large, subjects perceive two overlapping sheets of moving dots. This percept is called transparent motion. The occurrence of transparent motion cannot be explained by direction averaging, since that would result in a single direction of perceived motion.

Rather than just being a quirk of the human visual system, transparent motion is an important issue in motion processing. For example, when a robot is moving its

motion leads to a velocity field. The ability to detect transparent motion within that velocity field enables the robot to detect other moving objects at the same time that the velocity field can be used to estimate the heading direction of the robot. Without the ability to code multiple directions of motion at the same location, i.e. without the provision for transparent motion, this capacity is not available. Traditional algorithms have failed to properly process transparent motion, mainly because they assigned a unique velocity signal to each location, instead of allowing the possibility for multiple motion signals at a single location. Consequently, the study of transparent motion has recently enjoyed widespread interest.

## STIMULUS          PERCEPT

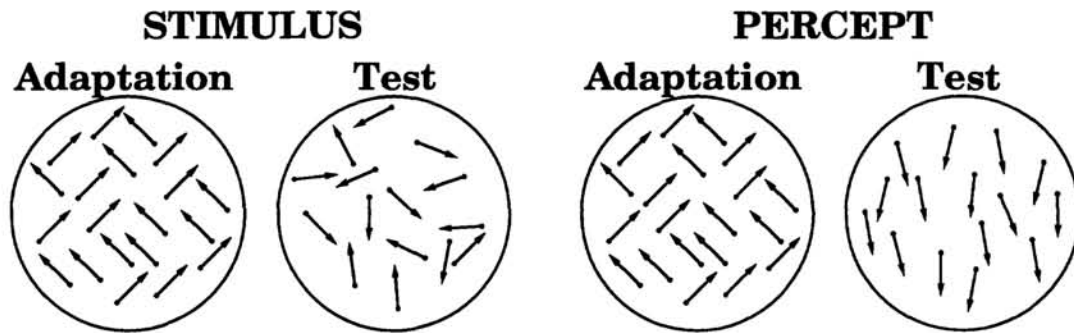

Figure 1: Two populations of dots moving in different directions during an adaptation phase are perceived as transparent motion. Subsequent viewing of randomly moving dots during a test phase leads to an illusory percept of unidirectional motion, the motion aftereffect (MAE). Stimulus and percept in both phases are shown.

After prolonged exposure to an *adaptation* display containing dots moving in one direction, randomly moving dots in a *test* display appear to be moving in the opposite direction (Hiris & Blake, 1992; Wohlgemuth, 1911). This illusory percept of motion is called the motion aftereffect (MAE). Traditionally this is explained by assuming that pairs of oppositely tuned direction selective units together code the presence of motion. When both are equally active, no motion is seen. Visual motion leads to stronger activation of one unit, and thus an imbalance in the activity of the two units. Consequently, motion is perceived. Activation of that unit causes it to fatigue, which means its response weakens. After motion offset, the previously active unit sends out a reduced signal compared to its partner due to adaptation. Thus adaptation generates an imbalance between the two units, and therefore illusory motion, the MAE, is perceived. This is the *ratio model* (Sutherland, 1961).

Recent psychophysical results show that after prolonged exposure to transparent motion, observers perceive a MAE of a single direction of motion, pointing in the vector average of the adaptation directions (Mather, 1980; Verstraten, Fredericksen, & van de Grind, 1994). Thus adaptation to transparent motion leads to a non-transparent MAE. This is illustrated in Figure 1. This result cannot be accounted for by the ratio model, since the non-transparent MAE does not point in the direction opposite to either of the adaptation directions. Instead, this result suggests that direction selective units of all directions interact and thus contribute to the MAE. This explanation is called the *distribution-shift model* (Mather, 1980). However, thus far it has only been vaguely defined, and no demonstration has been given that shows how this mechanism might work.

This study develops a model of human motion perception based on elements from both the ratio and the distribution-shift models for the MAE. The model is also applicable to the situation where two directions of motion are present. When the directions differ slightly, only a single direction is perceived. When the directions differ a lot, transparent motion is perceived. Both cases lead to a unitary MAE.

## 2   OUTLINE OF THE MODEL

The model consists of two stages. Both stages contain units that are direction selective. The architecture of the model is shown in Figure 2.

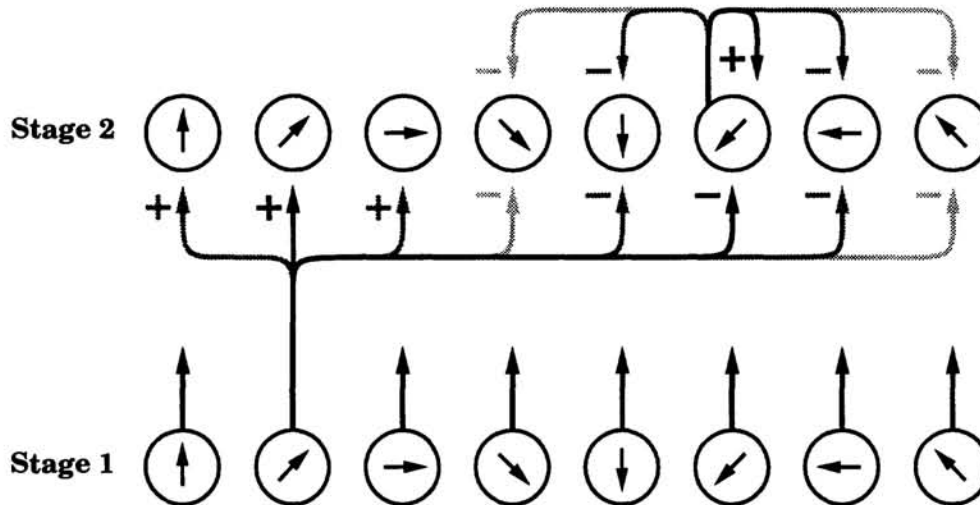

Figure 2: The model contains two stages of direction selective units. Units at stage 1 excite units of like direction selectivity at stage 2, and inhibit units of opposite directions. At stage 2 recurrent inhibition sharpens directional motion responses. The grey level indicates the strength of interaction between units. Strong influence is indicated by black arrows, weak influence is indicated by light grey arrows.

Units in stage 1 are broadly tuned motion detectors. In the present study the precise mechanism of motion detection is not central, and hence it has not been modeled. It is assumed that the bandwidth of motion detectors at this stage is about 30 degrees (Raymond, 1993; Williams, Tweten, & Sekuler, 1991). In the absence of any visual motion, all units are active at a baseline level; this is equivalent to neuronal noise. Whenever motion of a particular direction is present in the input, the activity of the corresponding unit $(v_i)$ is activated maximally $(v_i = 9)$, and units of similar direction selectivity are weakly activated $(v_i = 3)$. The activities of all other units decrease to zero. Associated with each unit $i$ at stage 1 is a weight $w_i$ that denotes the adaptational state of unit $i$ to fire a unit at stage 2. During prolonged exposure to motion these weights adapt, and their strength decreases. The equation governing the strength of the weights is given below:

$$\frac{dw_i}{dt} = R(1 - w_i) - v_i w_i,$$

where $R = 0.5$ denotes the rate of recovery to the baseline weight. When $w_i = 1$ the corresponding unit is not adapted. The further $w_i$ is reduced from 1, the more

the corresponding unit is adapted. The products $v_i w_i$ are transmitted to stage 2. Each unit of stage 1 excites units coding similar directions at stage 2, and inhibits units coding opposite directions of motion. The excitatory and inhibitory effects between units at stages 1 and 2 are caused by kernels, shown in Figure 3.

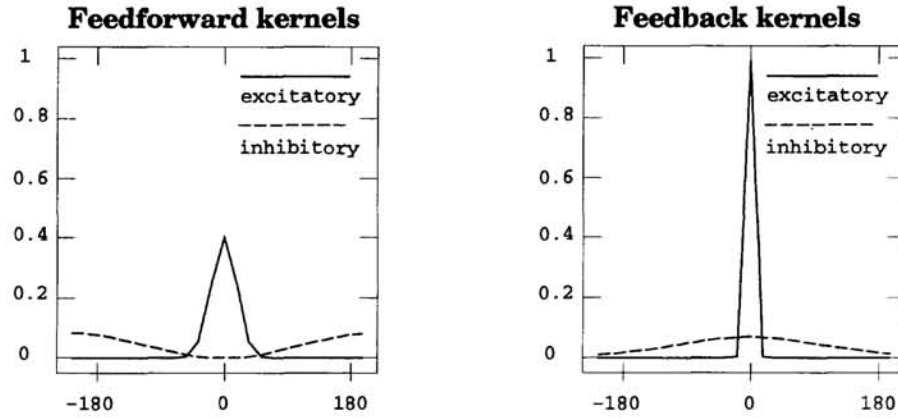

Figure 3: Kernels used in the model. Left: excitatory and inhibitory kernels between stages 1 and 2; right: excitatory and inhibitory feedback kernels within stage 2.

Activities at stage 2 are highly tuned for the direction of motion. The broad activation of motion signals at stage 1 is directionally sharpened at stage 2 through the interactions between recurrent excitation and inhibition. Each unit in stage 2 excites itself, and interacts with other units at stage 2 through recurrent inhibition. This inhibition is maximal for close directions, and falls off as the directions become more dissimilar. The kernels mediating excitatory and inhibitory interactions within stage 2 are shown in Figure 3. Through these inhibitory interactions the directional tuning of units at stage 2 is sharpened; through the excitatory feedback it is ensured that one unit will be maximally active. Activities of units at stage 2 are given by $M_i = \max^4(m_i, 0)$, where the behavior of $m_i$ is governed by:

$$\frac{dm_i}{dt} = -m_i + (1 - m_i)(F_i^+ + B_i^+) - (1 + m_i)(F_i^- + B_i^-).$$

$F_i^+$ and $F_i^-$ denote the result of convolving the products of the activities at stage 1 and the corresponding adaptation level, $v_j w_j$, with excitatory and inhibitory feedforward kernels respectively. Similarly, $B_i^+$ and $B_i^-$ denote the convolution of the activities $M_j$ at stage 2 with the feedback kernels.

## 3   SIMULATIONS OF PSYCHOPHYSICAL RESULTS

In the simulations there were 24 units at each stage. The model was simulated dynamically by integrating the differential equations using a fourth order Runge-Kutta method with stepsize $H = 0.01$ time units. The spacing of units in direction space was 15 degrees at both stages. Spatial interactions were not modeled. In the simulations shown, a motion stimulus is present until $t = 3$. Then the motion stimulus ceases. Activity at stage 2 after $t = 3$ corresponds to a MAE.

## 3.1  UNIDIRECTIONAL MOTION

When adapting to a single direction of motion, the model correctly generates a motion signal for that particular direction of motion. After offset of the motion input, the unit coding the opposite direction of motion is activated, as in the MAE. A simulation of this is shown in Figure 4.

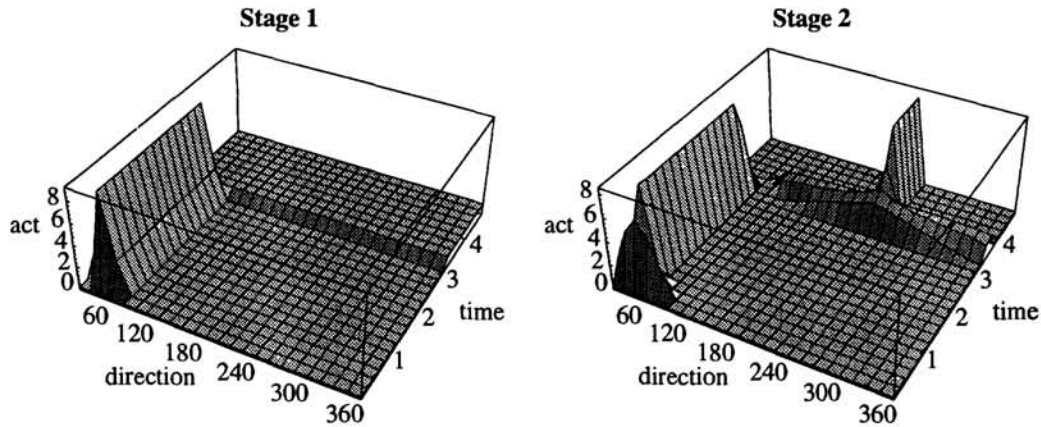

Figure 4: Simulation of single motion input and resulting MAE. Motion input is presented until $t = 3$.

During adaptation the motion stimulus excites the corresponding units at stage 1, which in turn activate units at stage 2. Due to recurrent inhibition only one unit at stage 2 remains active (Grossberg, 1973), and thus a very sharp motion signal is registered at stage 2. During adaptation the weights associated with the units that receive a motion input decrease. After motion offset, all units receive the same baseline input. Since the weights of the previously active units are decreased, the corresponding cells at stage 2 receive less feedforward excitation. At the same time, the previously active units receive strong feedforward inhibition, since they receive inhibition from units tuned to very different directions of motion and whose weights did not decay during adaptation. Similarly, the units coding the opposite direction of motion as those previously active receive more excitation and less inhibition. Through recurrent inhibition the unit at stage 2 coding the opposite direction to that which was active during adaptation is activated after motion offset: this activity corresponds to the MAE. Thus the MAE is primarily an effect of disinhibition.

## 3.2  TRANSPARENT MOTION: SIMILAR DIRECTIONS

Two populations of dots moving in different, but very similar, directions lead to bimodal activation at stage 1. Since the feedforward excitatory kernel is broadly tuned, and since the directions of motion are similar, the ensuing distribution of activities at stage 2 is unimodal, peaking halfway between the two directions of motion. This corresponds to the vector average of the directions of motion of the two populations of dots. A simulation of this is shown in Figure 5.

During adaptation the units at stage 1 corresponding to the input adapt. As before this means that after motion offset the previously active units receive less excitatory input and more inhibitory input. As during adaptation this signal is unimodal. Also, the unit at stage 2 coding the opposite direction to that of the stimulus receives

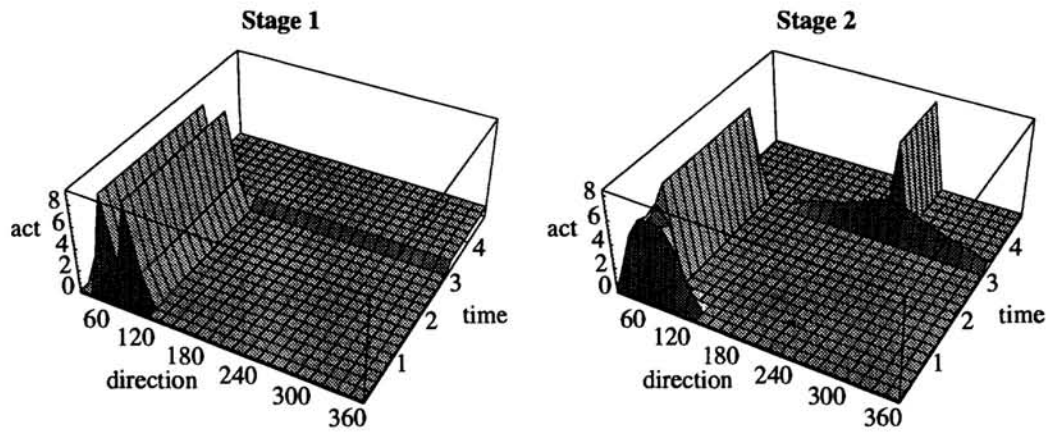

Figure 5: Simulation of two close directions of motion. Stage 2 of the network model registers unitary motion and a unitary MAE.

less inhibition and more excitation. Through the recurrent activities within stage 2, that unit gets maximally activated. A unimodal MAE results.

## 3.3   TRANSPARENT MOTION: DIFFERENT DIRECTIONS

When the directions of the two populations of dots in a transparent motion display are sufficiently distinct, the distribution of activities at stage 2 is no longer unimodal, but bimodal. Thus, recurrent inhibition leads to activation of two units at stage 2. They correspond to the two stimulus directions. A simulation is shown in Figure 6.

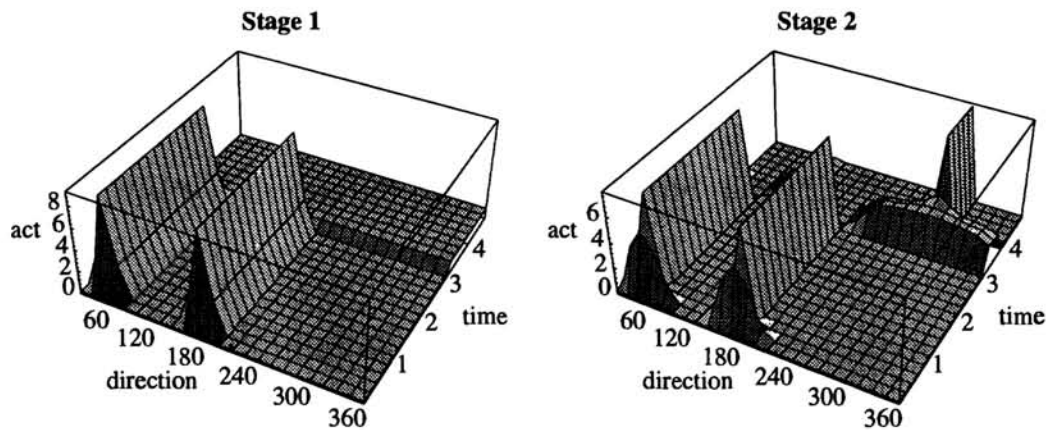

Figure 6: Simulation of two distinct directions of motion. Stage 2 of the model registers transparent motion during adaptation, but the MAE is unidirectional.

Feedforward inhibition is tuned much broader than feedforward excitation, and as a consequence the inhibitory signal during adaptation is unimodal, peaking at the unit of stage 2 coding the opposite direction of the average of the two previously active directions. Therefore that unit receives the least amount of inhibition after motion offset. It receives the same activity from stage 1 as units coding nearby directions, since the corresponding weights at stage 1 did not adapt. Due to recurrent activities at stage 2 that unit becomes active: non-transparent motion is registered.

# 4   DISCUSSION

Recently Snowden, Treue, Erickson, and Andersen (1991) have studied the effect of transparent motion stimuli on neurons in areas V1 and MT of macaque monkey. They simultaneously presented two populations of dots, one of which was moving in the preferred direction of the neuron under study, and the other population was moving in a different direction. They found that neurons in V1 were barely affected by the second population of dots. Neurons in MT, on the other hand, were inhibited when the direction of the second population differed from the preferred direction, and inhibition was maximal when the second population was moving opposite to the preferred direction. These results support key mechanisms of the model. At stage 1 there is no interaction between opposing directions of motion. The feedforward inhibition between stages 1 and 2 is maximal between opposite directions. Thus activities of units at stage 1 parallel neural activities recorded at V1, and activities of units at stage 2 parallels those neural activities recorded in area MT.

## Acknowledgments

This research was carried out under HFSP grant SF-354/94.

# Reference

Grossberg, S. (1973). Contour enhancement, short term memory, and constancies in reverberating neural networks. *Studies in Applied Mathematics, LII*, 213-257.

Hiris, E., & Blake, R. (1992). Another perspective in the visual motion aftereffect. *Proceedings of the National Academy of Sciences USA, 89*, 9025-9028.

Mather, G. (1980). The movement aftereffect and a distribution-shift model for coding the direction of visual movement. *Perception, 9*, 379-392.

Raymond, J. E. (1993). Movement direction analysers: independence and bandwidth. *Vision Research, 33*(5/6), 767-775.

Snowden, R. J., Treue, S., Erickson, R. G., & Andersen, R. A. (1991). The response of area MT and V1 neurons to transparent motion. *Journal of Neuroscience, 11*(9), 2768-2785.

Sutherland, N. S. (1961). Figural after-effects and apparent size. *Quarterly Journal of Experimental Psychology, 13*, 222-228.

Verstraten, F. A. J., Fredericksen, R. E., & van de Grind, W. A. (1994). Movement aftereffect of bi-vectorial transparent motion. *Vision Research, 34*, 349-358.

Williams, D., Tweten, S., & Sekuler, R. (1991). Using metamers to explore motion perception. *Vision Research, 31*(2), 275-286.

Williams, D. W., & Sekuler, R. (1984). Coherent global motion percept from stochastic local motions. *Vision Research, 24*(1), 55-62.

Wohlgemuth, A. (1911). On the aftereffect of seen movement. *British Journal of Psychology (Monograph Supplement), 1*, 1-117.

## Footnotes

* Present address: Caltech, Mail Code 216-76, Pasadena, CA 91125.
